# Instance-Specific Bayesian Model Averaging for Classification

**Shyam Visweswaran**
Center for Biomedical Informatics
Intelligent Systems Program
Pittsburgh, PA 15213
*shyam@cbmi.pitt.edu*

**Gregory F. Cooper**
Center for Biomedical Informatics
Intelligent Systems Program
Pittsburgh, PA 15213
*gfc@cbmi.pitt.edu*

## Abstract

Classification algorithms typically induce *population-wide* models that are trained to perform well on average on expected future instances. We introduce a Bayesian framework for learning *instance-specific* models from data that are optimized to predict well for a particular instance. Based on this framework, we present a lazy instance-specific algorithm called ISA that performs selective model averaging over a restricted class of Bayesian networks. On experimental evaluation, this algorithm shows superior performance over model selection. We intend to apply such instance-specific algorithms to improve the performance of patient-specific predictive models induced from medical data.

## 1   Introduction

Commonly used classification algorithms, such as neural networks, decision trees, Bayesian networks and support vector machines, typically induce a single model from a training set of instances, with the intent of applying it to all future instances. We call such a model a *population-wide model* because it is intended to be applied to an entire population of future instances. A population-wide model is optimized to predict well on average when applied to expected future instances. In contrast, an *instance-specific model* is one that is constructed specifically for a particular instance. The structure and parameters of an instance-specific model are specialized to the particular features of an instance, so that it is optimized to predict especially well for that instance.

Usually, methods that induce population-wide models employ *eager* learning in which the model is induced from the training data before the test instance is encountered. In contrast, *lazy* learning defers most or all processing until a response to a test instance is required. Learners that induce instance-specific models are necessarily lazy in nature since they take advantage of the information in the test instance. An example of a lazy instance-specific method is the *lazy Bayesian rule* (LBR) learner, implemented by Zheng and Webb [1], which induces rules in a lazy fashion from examples in the neighborhood of the test instance. A rule generated by LBR consists of a conjunction of the attribute-value pairs present in the test instance

as the antecedent and a local simple (naïve) Bayes classifier as the consequent. The structure of the local simple Bayes classifier consists of the attribute of interest as the parent of all other attributes that do not appear in the antecedent, and the parameters of the classifier are estimated from the subset of training instances that satisfy the antecedent. A greedy step-forward search selects the optimal LBR rule for a test instance to be classified. When evaluated on 29 UCI datasets, LBR had the lowest average error rate when compared to several eager learning methods [1].

Typically, both eager and lazy algorithms select a single model from some model space, ignoring the uncertainty in model selection. Bayesian model averaging is a coherent approach to dealing with the uncertainty in model selection, and it has been shown to improve the predictive performance of classifiers [2]. However, since the number of models in practically useful model spaces is enormous, exact model averaging over the entire model space is usually not feasible. In this paper, we describe a lazy instance-specific averaging (ISA) algorithm for classification that approximates Bayesian model averaging in an instance-sensitive manner. ISA extends LBR by adding Bayesian model averaging to an instance-specific model selection algorithm.

While the ISA algorithm is currently able to directly handle only discrete variables and is computationally more intensive than comparable eager algorithms, the results in this paper show that it performs well. In medicine, such lazy instance-specific algorithms can be applied to patient-specific modeling for improving the accuracy of diagnosis, prognosis and risk assessment.

The rest of this paper is structured as follows. Section 2 introduces a Bayesian framework for instance-specific learning. Section 3 describes the implementation of ISA. In Section 4, we evaluate ISA and compare its performance to that of LBR. Finally, in Section 5 we discuss the results of the comparison.

## 2   Decision Theoretic Framework

We use the following notation. Capital letters like $X$, $Z$, denote random variables and corresponding lower case letters, $x$, $z$, denote specific values assigned to them. Thus, $X = x$ denotes that variable $X$ is assigned the value $x$. Bold upper case letters, such as $\boldsymbol{X}$, $\boldsymbol{Z}$, represent sets of variables or random vectors and their realization is denoted by the corresponding bold lower case letters, $\boldsymbol{x}$, $\boldsymbol{z}$. Hence, $\boldsymbol{X} = \boldsymbol{x}$ denotes that the variables in $\boldsymbol{X}$ have the states given by $\boldsymbol{x}$. In addition, $Z$ denotes the target variable being predicted, $\boldsymbol{X}$ denotes the set of attribute variables, $M$ denotes a model, $D$ denotes the training dataset, and $<\boldsymbol{X^t}, Z^t>$ denotes a generic test instance that is not in $D$.

We now characterize population-wide and instance-specific model selection in decision theoretic terms. Given training data $D$ and a separate generic test instance $<\boldsymbol{X^t}, Z^t>$, the *Bayes optimal prediction* for $Z^t$ is obtained by combining the predictions of all models weighted by their posterior probabilities, as follows:

$$P(Z^t \mid \boldsymbol{X^t}, D) = \int_M P(Z^t \mid \boldsymbol{X^t}, M) P(M \mid D) dM \ . \tag{1}$$

The *optimal population-wide model* for predicting $Z^t$ is as follows:

$$\max_M \left\{ \sum_{\boldsymbol{X^t}} U\big[P(Z^t \mid \boldsymbol{X^t}, D), P(Z^t \mid \boldsymbol{X^t}, M)\big] P(\boldsymbol{X} \mid D) \right\}, \tag{2}$$

where the function $U$ gives the utility of approximating the Bayes optimal estimate $P(Z^t \mid X^t, D)$, with the estimate $P(Z^t \mid X^t, M)$ obtained from model $M$. The term $P(X \mid D)$ is given by:

$$P(X \mid D) = \int_M P(X \mid M)P(M \mid D)dM .$$  (3)

The *optimal instance-specific model* for predicting $Z^t$ is as follows:

$$\max_M \left\{ U\left[ P(Z^t \mid X^t = x^t, D), P(Z^t \mid X^t = x^t, M) \right] \right\},$$  (4)

where $x^t$ are the values of the attributes of the test instance $X^t$ for which we want to predict $Z^t$. The Bayes optimal estimate $P(Z^t \mid X^t = x^t, D)$, in Equation 4 is derived using Equation 1, for the special case in which $X^t = x^t$.

The difference between the population-wide and the instance-specific models can be noted by comparing Equations 2 and 4. Equation 2 for the population-wide model selects the model that on average will have the greatest utility. Equation 4 for the instance-specific model, however, selects the model that will have the greatest expected utility for the specific instance $X^t = x^t$. For predicting $Z^t$ in a given instance $X^t = x^t$, the model selected using Equation 2 can never have an expected utility greater than the model selected using Equation 4. This observation provides support for developing instance-specific models.

Equations 2 and 4 represent theoretical ideals for population-wide and instance-specific model selection, respectively; we are not suggesting they are practical to compute. The current paper focuses on model averaging, rather than model selection. Ideal Bayesian model averaging is given by Equation 1. Model averaging has previously been applied using population-wide models. Studies have shown that approximate Bayesian model averaging using population-wide models can improve predictive performance over population-wide model selection [2]. The current paper concentrates on investigating the predictive performance of approximate Bayesian model averaging using instance-specific models.

## 3   Instance-Specific Algorithm

We present the implementation of the lazy instance-specific algorithm based on the above framework. ISA searches the space of a restricted class of Bayesian networks to select a subset of the models over which to derive a weighted (averaged) posterior of the target variable $Z^t$. A key characteristic of the search is the use of a heuristic to select models that will have a significant influence on the weighted posterior. We introduce Bayesian networks briefly and then describe ISA in detail.

### 3.1   Bayesian Networks

A Bayesian network is a probabilistic model that combines a graphical representation (the Bayesian network structure) with quantitative information (the parameters of the Bayesian network) to represent the joint probability distribution over a set of random variables [3]. Specifically, a Bayesian network $M$ representing the set of variables $X$ consists of a pair $(G, \Theta_G)$. $G$ is a directed acyclic graph that contains a node for every variable in $X$ and an arc between every pair of nodes if the corresponding variables are directly probabilistically dependent. Conversely, the absence of an arc between a pair of nodes denotes probabilistic independence between the corresponding variables. $\Theta_G$ represents the parameterization of the model.

In a Bayesian network $M$, the immediate predecessors of a node $X_i$ in $X$ are called the parents of $X_i$ and the successors, both immediate and remote, of $X_i$ in $X$ are called the descendants of $X_i$. The immediate successors of $X_i$ are called the children of $X_i$. For each node $X_i$ there is a local probability distribution (that may be discrete or continuous) on that node given the state of its parents. The complete joint probability distribution over $X$, represented by the parameterization $\Theta_G$, can be factored into a product of local probability distributions defined on each node in the network. This factorization is determined by the independences captured by the structure of the Bayesian network and is formalized in the Bayesian network Markov condition: A node (representing a variable) is independent of its non-descendants given just its parents. According to this Markov condition, the joint probability distribution on model variables $X = (X_1, X_2, \ldots, X_n)$ can be factored as follows:

$$P(X_1, X_2, \ldots, X_n) = \prod_{i=1}^{n} P(X_i \mid parents(X_i)), \qquad (5)$$

where $parents(X_i)$ denotes the set of nodes that are the parents of $X_i$. If $X_i$ has no parents, then the set $parents(X_i)$ is empty and $P(X_i \mid parents(X_i))$ is just $P(X_i)$.

### 3.2 ISA Models

The LBR models of Zheng and Webb [1] can be represented as members of a restricted class of Bayesian networks (see Figure 1). We use the same class of Bayesian networks for the ISA models, to facilitate comparison between the two algorithms. In Figure 1, all nodes represent attributes that are discrete. Each node in $X$ has either an outgoing arc into target node, $Z$, or receives an arc from $Z$. That is, each node is either a parent or a child of $Z$. Thus, $X$ is partitioned into two sets: the first containing nodes ($X_1, \ldots, X_j$ in Figure 1) each of which is a parent of $Z$ and every node in the second set, and the second containing nodes ($X_{j+1}, \ldots, X_k$ in Figure 1) that have as parents the node $Z$ and every node in the first set. The nodes in the first set are instantiated to the corresponding values in the test instance for which $Z^t$ is to be predicted. Thus, the first set of nodes represents the antecedent of the LBR rule and the second set of nodes represents the consequent.

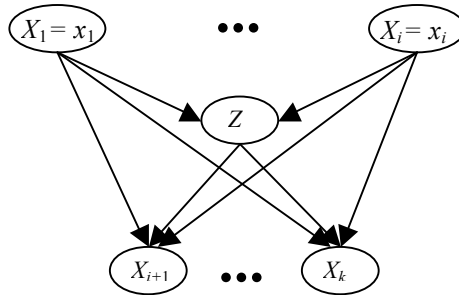

**Figure 1**: An example of a Bayesian network LBR model with target node $Z$ and $k$ attribute nodes of which $X_1, \ldots, X_j$ are instantiated to values $x_1, \ldots, x_j$ in $x^t$. $X_1, \ldots, X_j$ are present in the antecedent of the LBR rule and $Z, X_{j+1}, \ldots, X_k$ (that form the local simple Bayes classifier) are present in the consequent. The indices need not be ordered as shown, but are presented in this example for convenience of exposition.

### 3.3   Model Averaging

For Bayesian networks, Equation 1 can be evaluated as follows:

$$P(Z^t \mid \boldsymbol{x^t}, D) = \sum_M P(Z^t \mid \boldsymbol{x^t}, M) P(M \mid D), \tag{6}$$

with $M$ being a Bayesian network comprised of structure $G$ and parameters $\Theta_G$. The probability distribution of interest is a weighted average of the posterior distribution over all possible Bayesian networks where the weight is the probability of the Bayesian network given the data. Since exhaustive enumeration of all possible models is not feasible, even for this class of simple Bayesian networks, we approximate exact model averaging with selective model averaging. Let $R$ be the set of models selected by the search procedure from all possible models in the model space, as described in the next section. Then, with selective model averaging, $P(Z^t \mid \boldsymbol{x^t}, D)$ is estimated as:

$$P(Z^t \mid \boldsymbol{x^t}, D) \cong \frac{\sum_{M \in R} P(Z^t \mid \boldsymbol{x^t}, M) P(M \mid D)}{\sum_{M \in R} P(M \mid D)}. \tag{7}$$

Assuming uniform prior belief over all possible models, the model posterior $P(M \mid D)$ in Equation 7 can be replaced by the marginal likelihood $P(D \mid M)$, to obtain the following equation:

$$P(Z^t \mid \boldsymbol{x^t}, D) \cong \frac{\sum_{M \in R} P(Z^t \mid \boldsymbol{x^t}, M) P(D \mid M)}{\sum_{M \in R} P(D \mid M)}. \tag{8}$$

The (unconditional) marginal likelihood $P(D \mid M)$ in Equation 8, is a measure of the goodness of fit of the model to the data and is also known as the model score. While this score is suitable for assessing the model's fit to the joint probability distribution, it is not necessarily appropriate for assessing the goodness of fit to a conditional probability distribution which is the focus in prediction and classification tasks, as is the case here. A more suitable score in this situation is a conditional model score that is computed from training data $D$ of $d$ instances as:

$$score(D, M) = \prod_{p=1}^{d} P(z^p \mid \boldsymbol{x}^1, ..., \boldsymbol{x}^p, z^1, ..., z^{p-1}, M). \tag{9}$$

This score is computed in a predictive and sequential fashion: for the $p$[th] training instance the probability of predicting the observed value $z^p$ for the target variable is computed based on the values of all the variables in the preceding $p$-1 training instances and the values $\boldsymbol{x}^p$ of the attributes in the $p$[th] instance. One limitation of this score is that its value depends on the ordering of the data. Despite this limitation, it has been shown to be an effective scoring criterion for classification models [4].

The parameters of the Bayesian network $M$, used in the above computations, are defined as follows:

$$P(X_i = k \mid parents(X_i) = j) \equiv \theta_{ijk} = \frac{N_{ijk} + \alpha_{ijk}}{N_{ij} + \alpha_{ij}}, \tag{10}$$

where (i) $N_{ijk}$ is the number of instances in the training dataset $D$ where variable $X_i$ has value $k$ and the parents of $X_i$ are in state $j$, (ii) $N_{ij} = \sum_k N_{ijk}$, (iii) $\alpha_{ijk}$ is a

parameter prior that can be interpreted as the belief equivalent of having previously observed $\alpha_{ijk}$ instances in which variable $X_i$ has value $k$ and the parents of $X_i$ are in state $j$, and (iv) $\alpha_{ij} = \sum_k \alpha_{ijk}$ .

## 3.4 Model Search

We use a two-phase best-first heuristic search to sample the model space. The first phase ignores the evidence $\mathbf{x}^t$ in the test instance while searching for models that have high scores as given by Equation 9. This is followed by the second phase that searches for models having the greatest impact on the prediction of $Z^t$ for the test instance, which we formalize below.

The first phase searches for models that predict $Z$ in the training data very well; these are the models that have high conditional model scores. The initial model is the simple Bayes network that includes all the attributes in $\mathbf{X}$ as children of $Z$. A succeeding model is derived from a current model by reversing the arc of a child node in the current model, adding new outgoing arcs from it to $Z$ and the remaining children, and instantiating this node to the value in the test instance. This process is performed for each child in the current model. An incoming arc of a child node is considered for reversal only if the node's value is not missing in the test instance. The newly derived models are added to a priority queue, $Q$. During each iteration of the search, the model with the highest score (given by Equation 9) is removed from $Q$ and placed in a set $R$, following which new models are generated as described just above, scored and added to $Q$. The first phase terminates after a user-specified number of models have accumulated in $R$.

The second phase searches for models that change the current model-averaged estimate of $P(Z^t \mid \mathbf{x}^t, D)$ the most. The idea here is to find viable competing models for making this posterior probability prediction. When no competitive models can be found, the prediction becomes stable. During each iteration of the search, the highest ranked model $M^*$ is removed from $Q$ and added to $R$. The ranking is based on how much the model changes the current estimate of $P(Z^t \mid \mathbf{x}^t, D)$. More change is better. In particular, $M^*$ is the model in $Q$ that maximizes the following function:

$$f(R, M^*) = \left| g(R) - g(R \cup \{M^*\}) \right|, \tag{11}$$

where for a set of models $S$, the function $g(S)$ computes the approximate model averaged prediction for $Z^t$, as follows:

$$g(S) = \frac{\sum_{M \in S} P(Z^t \mid \mathbf{x}^t, M)\, score(D, M)}{\sum_{M \in S} score(D, M)} . \tag{12}$$

The second phase terminates when no new model can be found that has a value (as given by Equation 11) that is greater than a user-specified minimum threshold $T$. The final distribution of $Z^t$ is then computed from the models in $R$ using Equation 8.

## 4 Evaluation

We evaluated ISA on the 29 UCI datasets that Zheng and Webb used for the evaluation of LBR. On the same datasets, we also evaluated a simple Bayes classifier (SB) and LBR. For SB and LBR, we used the Weka implementations (Weka v3.3.6, http://www.cs.waikato.ac.nz/ml/weka/) with default settings [5]. We implemented the ISA algorithm as a standalone application in Java. The following

settings were used for ISA: a maximum of 100 phase-1 models, a threshold $T$ of 0.001 in phase-2, and an upper limit of 500 models in $R$. For the parameter priors in Equation 10, all $\alpha_{ijk}$ were set to 1.

All error rates were obtained by averaging the results from two stratified 10-fold cross-validation (20 trials total) similar to that used by Zheng and Webb. Since, both LBR and ISA can handle only discrete attributes, all numeric attributes were discretized in a pre-processing step using the entropy based discretization method described in [6]. For each pair of training and test folds, the discretization intervals were first estimated from the training fold and then applied to both folds. The error rates of two algorithms on a dataset were compared with a paired t-test carried out at the 5% significance level on the error rate statistics obtained from the 20 trials.

The results are shown in Table 1. Compared to SB, ISA has significantly fewer errors on 9 datasets and significantly more errors on one dataset. Compared to LBR, ISA has significantly fewer errors on 7 datasets and significantly more errors on two datasets. On two datasets, chess and tic-tac-toe, ISA shows considerable improvement in performance over both SB and LBR. With respect to computation

**Table 1**: Percent error rates of simple Bayes (SB), Lazy Bayesian Rule (LBR) and Instance-Specific Averaging (ISA). A - indicates that the ISA error rate is statistically significantly lower than the marked SB or LBR error rate. A + indicates that the ISA error rate is statistically significantly higher.

| Dataset | Size | No. of classes | Num. Attrib. | Nom. Attrib. | Percent error rate | | |
|---|---|---|---|---|---|---|---|
| | | | | | SB | LBR | ISA |
| Annealing | 898 | 6 | 6 | 32 | 3.5 - | 2.7 - | 1.9 |
| Audiology | 226 | 24 | 0 | 69 | 29.6 | 29.4 | 30.9 |
| Breast (W) | 699 | 2 | 9 | 0 | 2.9 + | 2.8 + | 3.7 |
| Chess (KR-KP) | 3169 | 2 | 0 | 36 | 12.1 - | 3.0 - | 1.1 |
| Credit (A) | 690 | 2 | 6 | 9 | 13.8 | 14.0 | 13.9 |
| Echocardiogram | 131 | 2 | 6 | 1 | 33.2 | 34.0 | 35.9 |
| Glass | 214 | 6 | 9 | 0 | 26.9 | 27.8 | 29.0 |
| Heart (C) | 303 | 2 | 13 | 0 | 16.2 | 16.2 | 17.5 |
| Hepatitis | 155 | 2 | 6 | 13 | 14.2 - | 14.2 - | 11.3 |
| Horse colic | 368 | 2 | 7 | 15 | 20.2 | 16.0 | 17.8 |
| House votes 84 | 435 | 2 | 0 | 16 | 10.1 - | 7.0 - | 5.1 |
| Hypothyroid | 3163 | 2 | 7 | 18 | 1.4 - | 0.9 | 0.9 |
| Iris | 150 | 3 | 4 | 0 | 6.0 | 6.0 | 5.3 |
| Labor | 57 | 2 | 8 | 8 | 8.8 | 6.1 | 7.0 |
| LED 24 | 200 | 10 | 0 | 24 | 40.5 | 40.5 | 40.3 |
| Liver disorders | 345 | 2 | 6 | 0 | 36.8 | 36.8 | 36.8 |
| Lung cancer | 32 | 3 | 0 | 56 | 56.3 | 56.3 | 56.3 |
| Lymphography | 148 | 4 | 0 | 18 | 15.5 - | 15.5 - | 13.2 |
| Pima | 768 | 2 | 8 | 0 | 21.8 | 22.0 | 22.3 |
| Postoperative | 90 | 3 | 1 | 7 | 33.3 | 33.3 | 33.3 |
| Primary tumor | 339 | 22 | 0 | 17 | 54.4 | 53.5 | 54.2 |
| Promoters | 106 | 2 | 0 | 57 | 7.5 | 7.5 | 7.5 |
| Solar flare | 1389 | 2 | 0 | 10 | 20.2 | 18.3 + | 19.4 |
| Sonar | 208 | 2 | 60 | 0 | 15.4 | 15.6 | 15.9 |
| Soybean | 683 | 19 | 0 | 35 | 7.9 - | 7.1 | 7.2 |
| Splice junction | 3177 | 3 | 0 | 60 | 4.7 | 4.3 | 4.4 |
| Tic-Tac-Toe | 958 | 2 | 0 | 9 | 30.3 - | 13.7 - | 10.3 |
| Wine | 178 | 3 | 13 | 0 | 1.1 | 1.1 | 1.1 |
| Zoo | 101 | 7 | 0 | 16 | 8.4 - | 8.4 - | 6.4 |

times, ISA took 6 times longer to run than LBR on average for a single test instance on a desktop computer with a 2 GHz Pentium 4 processor and 3 GB of RAM.

# 5   Conclusions and Future Research

We have introduced a Bayesian framework for instance-specific model averaging and presented ISA as one example of a classification algorithm based on this framework. An instance-specific algorithm like LBR that does model selection has been shown by Zheng and Webb to perform classification better than several eager algorithms [1]. Our results show that ISA, which extends LBR by adding Bayesian model averaging, improves overall on LBR, which provides support that we can obtain additional prediction improvement by performing instance-specific model averaging rather than just instance-specific model selection.

In future work, we plan to explore further the behavior of ISA with respect to the number of models being averaged and the effect of the number of models selected in each of the two phases of the search. We will also investigate methods to improve the computational efficiency of ISA. In addition, we plan to examine other heuristics for model search as well as more general model spaces such as unrestricted Bayesian networks.

The instance-specific framework is not restricted to the Bayesian network models that we have used in this investigation. In the future, we plan to explore other models using this framework. Our ultimate interest is to apply these instance-specific algorithms to improve patient-specific predictions (for diagnosis, therapy selection, and prognosis) and thereby to improve patient care.

## Acknowledgments

This work was supported by the grant T15-LM/DE07059 from the National Library of Medicine (NLM) to the University of Pittsburgh's Biomedical Informatics Training Program. We would like to thank the three anonymous reviewers for their helpful comments.

## References

[1] Zheng, Z. and Webb, G.I. (2000). Lazy Learning of Bayesian Rules. *Machine Learning*, 41(1):53-84.

[2] Hoeting, J.A., Madigan, D., Raftery, A.E. and Volinsky, C.T. (1999). Bayesian Model Averaging: A Tutorial. *Statistical Science*, 14:382-417.

[3] Pearl, J. (1988). *Probabilistic Reasoning in Intelligent Systems*. Morgan Kaufmann, San Mateo, CA.

[4] Kontkanen, P., Myllymaki, P., Silander, T., and Tirri, H. (1999). On Supervised Selection of Bayesian Networks. In *Proceedings of the 15th International Conference on Uncertainty in Artificial Intelligence*, pages 334-342, Stockholm, Sweden. Morgan Kaufmann.

[5] Witten, I.H. and Frank, E. (2000). *Data Mining: Practical Machine Learning Tools with Java Implementations*. Morgan Kaufmann, San Francisco, CA.

[6] Fayyad, U.M., and Irani, K.B. (1993). Multi-Interval Discretization of Continuous-Valued Attributes for Classification Learning. In *Proceedings of the Thirteenth International Joint Conference on Artificial Intelligence*, pages 1022-1027, San Mateo, CA. Morgan Kaufmann.
